# Distributional Population Codes and Multiple Motion Models

**Richard S. Zemel**
University of Arizona
zemel@u.arizona.edu

**Peter Dayan**
Gatsby Computational Neuroscience Unit
dayan@gatsby.ucl.ac.uk

## Abstract

Most theoretical and empirical studies of population codes make the assumption that underlying neuronal activities is a unique and unambiguous value of an encoded quantity. However, population activities can contain additional information about such things as multiple values of or uncertainty about the quantity. We have previously suggested a method to recover extra information by treating the activities of the population of cells as coding for a complete distribution over the coded quantity rather than just a single value. We now show how this approach bears on psychophysical and neurophysiological studies of population codes for motion direction in tasks involving transparent motion stimuli. We show that, unlike standard approaches, it is able to recover multiple motions from population responses, and also that its output is consistent with both correct and erroneous human performance on psychophysical tasks.

A population code can be defined as a set of units whose activities collectively encode some underlying variable (or variables). The standard view is that population codes are useful for accurately encoding the underlying variable when the individual units are noisy. Current statistical approaches to interpreting population activity reflect this view, in that they determine the optimal single value that explains the observed activity pattern given a particular model of the noise (and possibly a loss function).

In our work, we have pursued an alternative hypothesis, that the population encodes additional information about the underlying variable, including *multiple values* and *uncertainty*. The **Distributional Population Coding** (DPC) framework finds the best probability distribution across values that fits the population activity (Zemel, Dayan, & Pouget, 1998).

The DPC framework is appealing since it makes clear how extra information can be conveyed in a population code. In this paper, we use it to address a particu-

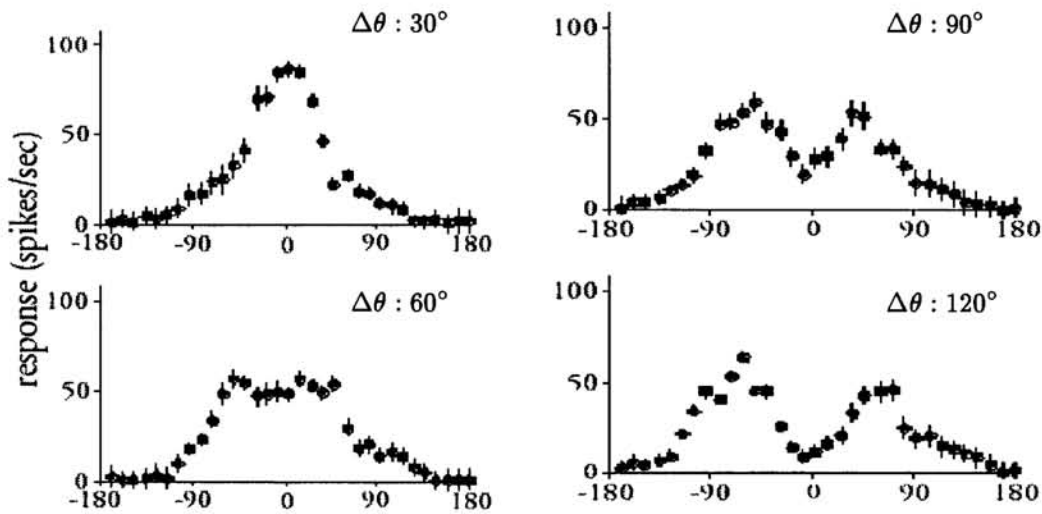

Figure 1: Each of the four plots depicts a single MT cell response (spikes per second) to a transparent motion stimulus of a fixed directional difference ($\Delta\theta$) between the two motion directions. The x-axis gives the average direction of stimulus motion relative to the cell's preferred direction ($0°$). From Treue, personal communication.

lar body of experimental data on transparent motion perception, due to Treue and colleagues (Hol & Treue, 1997; Rauber & Treue, 1997). These transparent motion experiments provide an ideal test of the DPC framework, in that the neurophysiological data reveal how the population responds to multiple values in the stimuli, and the psychophysical data describe how these values are actually decoded, putatively from the population response. We investigate how standard methods fare on these data, and compare their performance to that of DPC.

## 1   RESPONSES TO MULTIPLE MOTIONS

Many investigators have examined neural and behavioral responses to stimuli composed of two patterns sliding across each other. These often create the impression of two separate surfaces moving in different directions. The general neurophysiological finding is that an MT cell's response to these stimuli can be characterized as the average of its responses to the individual components (van Wezel et al., 1996; Recanzone et al., 1997). As an example, Figure 1 shows data obtained from single-cell recordings in MT to random dot patterns consisting of two distinct motion directions (Treue, personal communication). Each plot is for a different relative angle ($\Delta\theta$) between the two directions. A plot can equivalently be viewed as the response of an population of MT cells having different preferred directions to a single presentation of a stimulus containing two directions. If $\Delta\theta$ is large, the activity profile is bimodal, but as the directional difference shrinks, the profile becomes unimodal. The population response to a $\Delta\theta = 30°$ motion stimulus is merely a wider version of the response to a stimulus containing a single direction of motion. However, this transition from a bimodal to unimodal profiles in MT does not apparently correspond to subjects' percepts; subjects can reliably perceive both motions in superimposed transparent random patterns down to an angle of $10°$ (Mather & Moulden, 1983). If these MT activities play a determining role in motion perception, the challenge is to understand how the visual system can extract

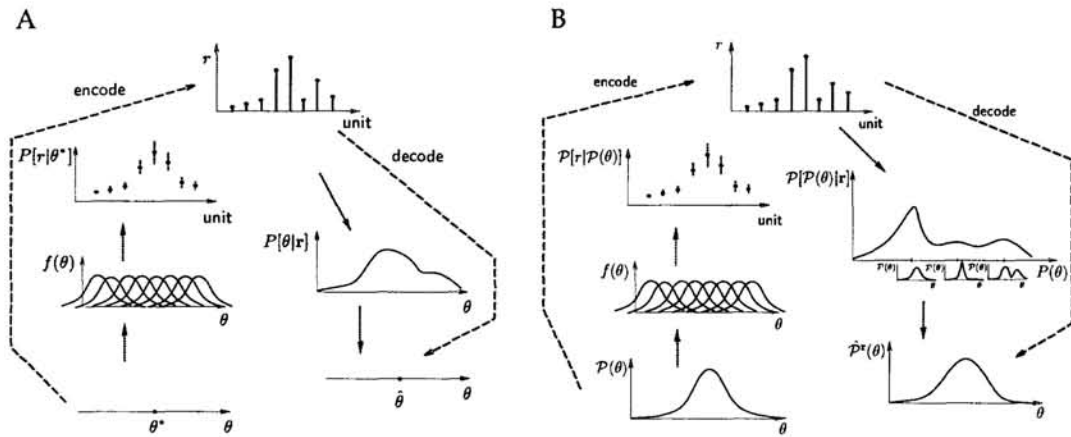

Figure 2: (A) The standard Bayesian population coding framework assumes that a single value is encoded in a set of noisy neural activities. (B) The distributional population coding framework shows how a distribution over $\theta$ can be encoded and then decoded from noisy population activities. From Zemel et al. (1998).

both motions from such unimodal (and bimodal) response profiles.

## 2   ENCODING & DECODING

Statistical population code decoding methods begin with the knowledge, collected over many experimental trials, of the tuning function $f_i(\theta)$ for each cell $i$, determined using simple stimuli (e.g., ones containing uni-directional motion). Figure 2A cartoons the framework used for standard decoding. Starting on the bottom left, encoding consists of taking a value $\theta$ to be coded and representing it by the noisy activities $r_i$ of the elements of a population code. In the simulations described here, we have used a population of 200 model MT cells, with tuning functions defined by random sampling within physiologically-determined ranges for the parameters: baseline $b$, amplitude $a$ and width $\sigma$. The encoding model comes from the MT data: for a single motion, $\langle r_i|\theta \rangle = f_i(\theta) = b_i + a_i \times \exp[-(\theta-\theta_i)^2/2\sigma_i^2]$, while for two motions, $\langle r_i|\theta_1,\theta_2 \rangle = \frac{1}{2}[f_i(\theta_1) + f_i(\theta_2)]$. The noise is taken to be independent and Poisson.

Standard Bayesian decoding starts with the activities $\mathbf{r} = \{r_i\}$ and generates a distribution $\mathcal{P}[\theta|\mathbf{r}]$. Under the model with Poisson noise,

$$\mathcal{P}[\theta|\mathbf{r}] \sim \log\left\{ \mathcal{P}[\theta] \prod_i \mathcal{P}[r_i|\theta] \right\} \sim \sum_i r_i \log f_i(\theta)$$

This method thus provides a multiplicative kernel density estimate, tending to produce a sharp distribution for a single motion direction $\theta$. A single estimate $\hat{\theta}$ can be extracted from $\mathcal{P}[\theta|\mathbf{r}]$ using a loss function.

For this method to decode successfully when there are two motions in the input ($\theta_1$ and $\theta_2$), the extracted distribution must at least have two modes. Standard Bayesian decoding fails to satisfy this requirement. First, if the response profile $\mathbf{r}$ is unimodal (cf. the 30° plot in Figure 1), convolution with unimodal kernels $\{\log f_i(\theta)\}$ produces a unimodal $\log \mathcal{P}[\theta|\mathbf{r}]$, peaked about the average of the two

directions. The additive kernel density estimate, an alternative distributional decoding method proposed by Anderson (1995), suffers from the same problem, and also fails to be adequately sharp for single value inputs.

Surprisingly, the standard Bayesian decoding method also fails on bimodal response profiles. If the baseline response $b_i = 0$, then $P[\theta|\mathbf{r}]$ is Gaussian, with mean $\sum_i r_i \theta_i / \sum_{i'} r_{i'}$ and variance $1/\sum_i r_i/\sigma_i^2$ (Snippe, 1996; Zemel et al., 1998). If $b_i > 0$, then, for the extracted distribution to have two modes in the appropriate positions, $\log[P[\theta_1|\mathbf{r}]/P[\theta_2|\mathbf{r}]]$ must be small. However, the variance of this quantity is $\sum_i \langle r_i \rangle \left( \log[f_i(\theta_1)/f_i(\theta_2)] \right)^2$, which is much greater than 0 unless the tuning curves are so flat as to be able to convey only little information about the stimuli. Intuitively, the noise in the rates causes $\sum r_i \log f_i(\theta)$ to be greater around one of the two values, and exponentiating to form $P[\theta|\mathbf{r}]$ selects out this one value. Thus the standard method can only extract one of the two motion components from the population responses to transparent motion.

The distributional population coding method (Figure 2B) extends the standard encoding model to allow $\mathbf{r}$ to depend on general $P[\theta]$:

$$\langle r_i \rangle = \int_\theta P[\theta] f_i(\theta) d\theta \tag{1}$$

Bayesian decoding takes the observed activities $\mathbf{r}$ and produces probability distributions over probability distributions over $\theta$, $P[P(\theta)|\mathbf{r}]$. For simplicity, we decode using an approximate form of maximum likelihood in distributions over $\theta$, finding the $\hat{P}^{\mathbf{r}}(\theta)$ that maximizes $L[P(\theta)|\mathbf{r}] \sim \sum_i r_i \log[f_i(\theta) * P(\theta)] - \alpha g[P(\theta)]$ where the smoothness term $g[]$ acts as a regularizer.

The distributional *encoding* operation in Equation 1 is quite straightforward – by design, since this represents an assumption about what neural processing prior to (in this case) MT performs. However, the distributional *decoding* operation that we have used (Zemel et al., 1998) involves complicated and non-neural operations. The idea is to understand what information in principle may be conveyed by a population code under this interpretation, and then to judge actual neural operations in the light of this theoretical optimum. DPC is a statistical cousin of so-called line-element models, which attempt to account for subjects' performance in cases like transparency using the output of some fixed number of direction-selective mechanisms (Williams et al., 1991).

## 3 DECODING MULTIPLE MOTIONS

We have applied our model to simulated MT response patterns $\mathbf{r}$ generated via the DPC encoding model (Equation 1). For multiple motion stimuli, with $P(\theta) = (\delta(\theta - \theta_1) + \delta(\theta - \theta_2))/2$, this encoding model produces the observed neurophysiological response: each unit's expected activity is the average of its responses to the component motions. For bimodal response patterns, DPC matches the generating distribution (Figure 3). For unimodal response patterns, such as those generated by double motion stimuli with $\Delta\theta = 30°$, DPC also consistently recovers the generating distribution. The bimodality of the reconstructed distribution begins to break down around $\Delta\theta = 10°$, which is also the point at which subjects are unable distinguish two motions from a single broader band of motion directions (Mather & Moulden, 1983).

It has been reported (Treue, personal communication) that for angles $\Delta\theta < 10°$, subjects can tell that all points are not moving in parallel, but are uncertain whether

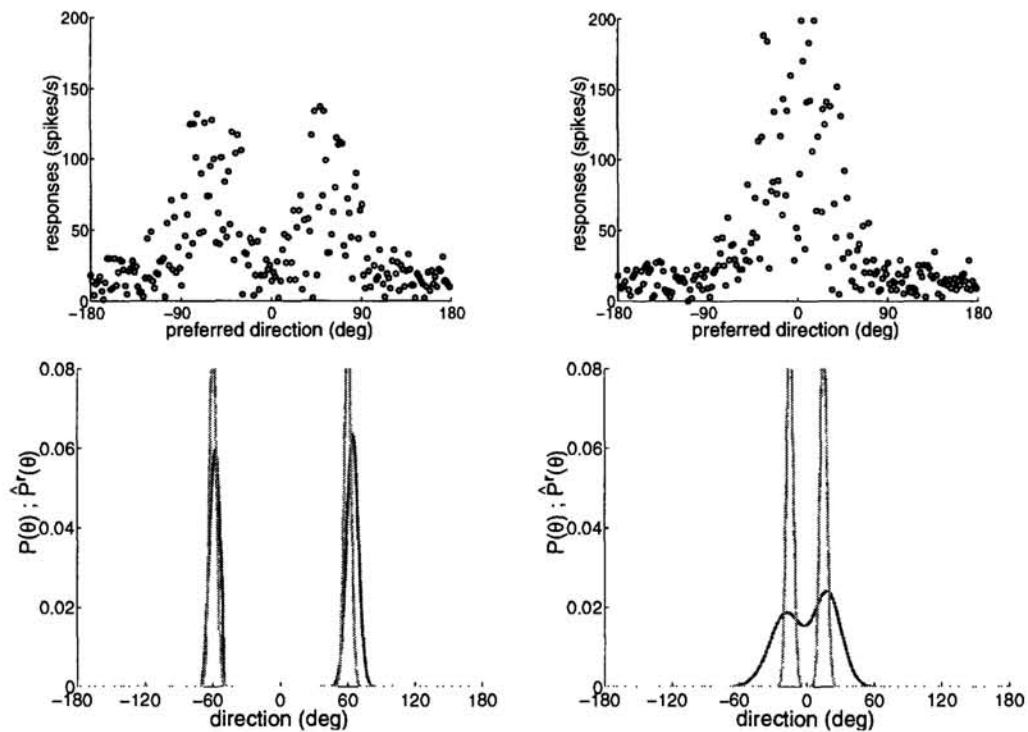

Figure 3: (A) On a single simulated trial, the population response forms a bi-modal activity profile when $\Delta\theta = 120°$. (B) The reconstructed (darker) distribution closely matches the true input distribution for this trial. (C) As $\Delta\theta \to 10°$, the population response is no longer bimodal, instead has a noisy unimodal profile, and (D) the reconstructed distribution no longer has two clear modes.

they are moving in two discrete directions or within a directional band. Our model qualitatively captures this uncertainty, reconstructing a broad distribution with two small peaks for directional differences between 7° and 10°.

DPC also matches psychophysical performance on metameric stimuli. Rauber and Treue (1997) asked human subjects to report the directions in moving dot patterns consisting of 2, 3 or 5 directions of motion. The motion directions were -40° and +40°; -50°, 0° and +50°; and -50°, -30°, 0°, +30°, and +50°, respectively, but the proportions of dots moving in each direction were adjusted so that the population responses produced by an encoding model similar to Equation 1 would all be the same. Subjects reported the same two motion directions, at -40° and 40°, to all three types of stimuli.

DPC, like any reasonably deterministic decoding model, takes these (essentially identical) patterns of activity and, metamerically, reports the same answer for each case. Unlike most models, its answer—that there are two motions at roughly ±40°—matches human responses. The *fact* of metamerization is not due to any kind of prior in the model as to the number of directions to be recovered. However, that the actual report in each case includes just two motions (when clearly three or five motions would be equally consistent with the input) is a consequence of the smoothness prior. We can go further with DPC and predict how changing the proportion of dots moving in the central of three directions would lead to different percepts – from a single motion to two as this proportion decreases.

We can further evaluate the performance of DPC by comparing the quality of its

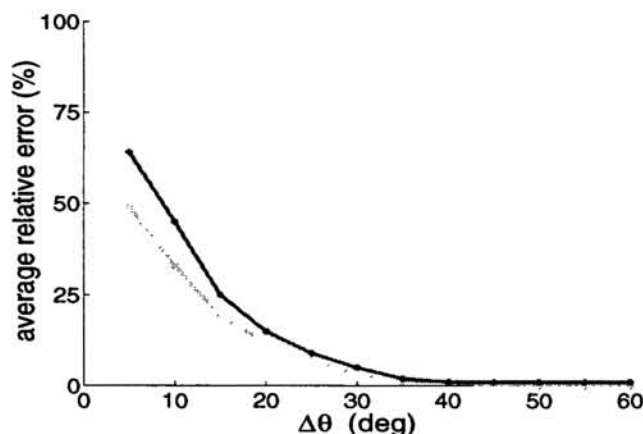

Figure 4: The average relative error $E$ in direction judgments (Equation 2) for the DPC model (top curve) and for a model with the correct prior for this particular input set.

reconstruction to that obtained by fitting the correct model of the input distribution, a mixture of delta functions. We simulated MT responses to motion stimuli composed of two evenly-weighted directions, with 100 examples for each value of $\Delta\theta$ in a range from $5°$ to $60°$. We fit a mixture of two delta functions to each population response, and measured the average relative error in direction judgments based on this fitted distribution versus the two true directions, $\theta_1$ and $\theta_2$ on that example $t$:

$$E = \frac{|\hat{\theta}_1^t - \theta_1| + |\hat{\theta}_2^t - \theta_2|}{2\Delta\theta} \qquad (2)$$

We then applied the DPC model to the same population codes. To measure the average error, we first fit the general distribution $\hat{\mathcal{P}}^r(\theta)$ produced by DPC with a pair of equal-weighted Gaussians, and determined $\hat{\theta}_1^t$ and $\hat{\theta}_2^t$ from the appropriate mean and variance. As can be seen in Figure 4, the DPC model, which only has a general smoothness prior over the form of the input distribution, preserves the information in the observed rates nearly as well as the model with the correct prior.

## 4  CONCLUSIONS

Transparent motion provides an ideal test of distributional population coding, since the encoding model is determined by neural activity and the decoding model by the behavioral data. Two existing kernel density estimate models, involving additive (Anderson, 1995) and multiplicative (standard Bayesian decoding) combination, perform poorly in this paradigm. DPC, a model in which neuronal responses and the animal's judgments are treated as being sensitive to the entire distribution of an encoded value, has been shown to be consistent with both single-cell responses and behavioral decisions, even matching subjects' threshold behavior.

We are currently applying this same model to several other motion experiments, including one in which subjects had to determine whether a motion stimulus consisted of a number of discrete directions or a uniform distribution (Williams et al., 1991). We are investigating whether our model can explain the nonmonotonic relationship between the number of directions and the judgments. We have also applied DPC to a notorious puzzle for population coding: that *single* MT cells are

just as accurate as the whole monkey – one cell's output could directly support inference of the same quality as the monkeys. Our approach provides an alternative explanation for part of this apparent inefficiency to that of the noisy pooling model of Shadlen et al. (1996). Finally, experiments showing the effect of target uncertainty on population responses (Basso & Wurtz, 1998; Bastian et al,. 1998) are also handled naturally by the DPC approach.

The current model is intended to describe the information available at one stage in the processing stream. It does not address the precise mechanism of motion encoding, i.e., how responses in MT arise. We also have not considered the neural decoding and decision mechanisms. These could likely involve a layer of units that reaches decisions through a pattern of feedforward and lateral connections, as in the model proposed by Grunewald (1996) for the detection of transparent motion.

One critical issue that remains is normalization. It is not clear how to distinguish ambiguity about a *single* value for the encoded variable from the existence of *multiple* values of that variable (as in transparency for motion). Various factors are likely to be important, including the degree of separation of the modes and also prior expectations about the possibility of equivalents of transparency.

Acknowledgements: This work was funded by ONR Young Investigator Award N00014-98-1-0509 to RZ, and NIMH grant 1R29MH5541-01, and grants from the Surdna Foundation and the Gatsby Charitable Foundation to PD. We thank Stefan Treue for providing us with the data plot and for informative discussions of his experiments; Alexandre Pouget and Charlie Anderson for useful discussions of distributed coding and the standard model; and Zoubin Ghahramani and Geoff Hinton for helpful conversations about reconstruction in the log probability domain.

# References

[1] Anderson, C. H. (1995). Unifying perspectives on neuronal codes and processing. In *XIX International workshop on condensed matter theories*. Caracas, Venezuela.

[2] Basso, M. A. & Wurtz, R. H. (1998). Modulation of neuronal activity in superior colliculus by changes in target probability. *Journal of Neuroscience, 18*(18), 7519-34.

[3] Bastian, A., Riehle, A., Erlhagen, W., & Schoner, G. (1998). Prior information preshapes the population representation of movement direction in motor cortex. *Neuroreport, 9*(2), 315-319.

[4] Britten, K. H., Shadlen, M. N., Newsome, W. T., & Movshon, J. A. (1992). The analysis of visual motion: A comparison of neuronal and psychophysical performance. *Journal of Neuroscience, 12*(12), 4745–4765.

[5] Grunewald, A. (1996). A model of transparent motion and non-transparent motion aftereffects. In D. S. Touretzky, M. C. Mozer, & M. E. Hasselmo (Eds.), *Advances in Neural Information Processing Systems 8* (pp. 837–843). Cambridge, MA: MIT Press.

[6] Hol, K. & Treue, S. (1997). Direction-selective responses in the superior temporal sulcus to transparent patterns moving at acute angles. *Society for Neuroscience Abstracts 23* (p. 179:11).

[7] Mather, G. & Moulden, B. (1983). Thresholds for movement direction: two directions are less detectable than one. *Quarterly Journal of Experimental Psychology, 35*, 513-518.

[8] Rauber, H. J. & Treue, S. (1997). Recovering the directions of visual motion in transparent patterns. *Society for Neuroscience Abstracts 23* (p. 179:10).

[9] Recanzone, G. H., Wurtz, R. H., & Schwarz, U. (1997). Responses of MT and MST neurons to one and two moving objects in the receptive field. *Journal of Neurophysiology, 78*(6), 2904–2915.

[10] Shadlen, M. N., Britten, K. H., Newsome, W. T., & Movshon, J. A. (1996). A computational analysis of the relationship between neuronal and behavioral responses to visual motion. *Journal of Neuroscience, 16*(4), 1486–510.

[11] Snippe, H. P. (1996). Theoretical considerations for the analysis of population coding in motor cortex. *Neural Computation, 8*(3):29–37.

[12] van Wezel, R. J., Lankheet, M. J., Verstraten, F. A., Maree, A. F., & van de Grind, W. A. (1996). Responses of complex cells in area 17 of the cat to bi-vectorial transparent motion. *Vision Research, 36*(18), 2805-13.

[13] Williams, D., Tweten, S., & Sekuler, R. (1991). Using metamers to explore motion perception. *Vision Research, 31*(2), 275–286.

[14] Zemel, R. S., Dayan, P., & Pouget, A. (1998). Probabilistic interpretation of population codes. *Neural Computation, 10*, 403–430.
